# Approximate Solutions to Optimal Stopping Problems

**John N. Tsitsiklis and Benjamin Van Roy**
Laboratory for Information and Decision Systems
Massachusetts Institute of Technology
Cambridge, MA 02139
e-mail: jnt@mit.edu, bvr@mit.edu

## Abstract

We propose and analyze an algorithm that approximates solutions to the problem of optimal stopping in a discounted irreducible aperiodic Markov chain. The scheme involves the use of linear combinations of fixed basis functions to approximate a $Q$–function. The weights of the linear combination are incrementally updated through an iterative process similar to $Q$–learning, involving simulation of the underlying Markov chain. Due to space limitations, we only provide an overview of a proof of convergence (with probability 1) and bounds on the approximation error. This is the first theoretical result that establishes the soundness of a $Q$–learning–like algorithm when combined with arbitrary linear function approximators to solve a sequential decision problem. Though this paper focuses on the case of finite state spaces, the results extend naturally to continuous and unbounded state spaces, which are addressed in a forthcoming full-length paper.

## 1   INTRODUCTION

Problems of sequential decision–making under uncertainty have been studied extensively using the methodology of dynamic programming [Bertsekas, 1995]. The hallmark of dynamic programming is the use of a *value function*, which evaluates expected future reward, as a function of the current state. Serving as a tool for predicting long-term consequences of available options, the value function can be used to generate optimal decisions.

A number of algorithms for computing value functions can be found in the dynamic programming literature. These methods compute and store one value per state in a state space. Due to the curse of dimensionality, however, states spaces are typically

intractable, and the practical applications of dynamic programming are severely limited.

The use of function approximators to "fit" value functions has been a central theme in the field of reinforcement learning. The idea here is to choose a function approximator that has a tractable number of parameters, and to tune the parameters to approximate the value function. The resulting function can then be used to approximate optimal decisions.

There are two preconditions to the development an effective approximation. First, we need to choose a function approximator that provides a "good fit" to the value function for some setting of parameter values. In this respect, the choice requires practical experience or theoretical analysis that provides some rough information on the shape of the function to be approximated. Second, we need effective algorithms for tuning the parameters of the function approximator.

Watkins (1989) has proposed the $Q$–learning algorithm as a possibility. The original analyses of Watkins (1989) and Watkins and Dayan (1992), the formal analysis of Tsitsiklis (1994), and the related work of Jaakkola, Jordan, and Singh (1994), establish that the algorithm is sound when used in conjunction with exhaustive look-up table representations (i.e., without function approximation). Jaakkola, Singh, and Jordan (1995), Tsitsiklis and Van Roy (1996a), and Gordon (1995), provide a foundation for the use of a rather restrictive class of function approximators with variants of $Q$–learning. Unfortunately, there is no prior theoretical support for the use of $Q$–learning–like algorithms when broader classes of function approximators are employed.

In this paper, we propose a variant of $Q$–learning for approximating solutions to optimal stopping problems, and we provide a convergence result that established its soundness. The algorithm approximates a $Q$–function using a linear combination of arbitrary fixed basis functions. The weights of these basis functions are iteratively updated during the simulation of a Markov chain. Our result serves as a starting point for the analysis of $Q$–learning–like methods when used in conjunction with classes of function approximators that are more general than piecewise constant. In addition, the algorithm we propose is significant in its own right. Optimal stopping problems appear in practical contexts such as financial decision making and sequential analysis in statistics. Like other problems of sequential decision making, optimal stopping problems suffer from the curse of dimensionality, and classical dynamic programming methods are of limited use. The method we propose presents a sound approach to addressing such problems.

## 2  OPTIMAL STOPPING PROBLEMS

We consider a discrete-time, infinite-horizon, Markov chain with a finite state space $S = \{1, \ldots, n\}$ and a transition probability matrix $P$. The Markov chain follows a trajectory $x_0, x_1, x_2, \ldots$ where the probability that the next state is $y$ given that the current state is $x$ is given by the $(x, y)$th element of $P$, and is denoted by $p_{xy}$. At each time $t \in \{0, 1, 2, \ldots\}$ the trajectory can be stopped with a terminal reward of $G(x_t)$. If the trajectory is not stopped, a reward of $g(x_t)$ is obtained. The objective is to maximize the expected infinite-horizon discounted reward, given by

$$E\left[\sum_{t=0}^{\tau-1} \alpha^t g(x_t) + \alpha^\tau G(x_\tau)\right],$$

where $\alpha \in (0, 1)$ is a discount factor and $\tau$ is the time at which the process is stopped. The variable $\tau$ is defined by a stopping policy, which is given by a sequence

of mappings $\mu_t : S^{t+1} \mapsto \{\text{stop}, \text{continue}\}$. Each $\mu_t$ determines whether or not to terminate, based on $x_0, \ldots, x_t$. If the decision is to terminate, then $\tau = t$.

We define the value function to be a mapping from states to the expected discounted future reward, given that an optimal policy is followed starting at a given state. In particular, the value function $J^* : S \mapsto \Re$ is given by

$$J^*(x) = \inf_{\{\mu_1, \mu_2, \ldots\}} E \left[ \sum_{t=0}^{\tau-1} \alpha^t g(x_t) + \alpha^\tau G(x_\tau) | x_0 = x \right],$$

where $\tau$ is the stopping time given by the policy $\{\mu_t\}$. It is well known that the value function is the unique solution to Bellman's equation:

$$J^*(x) = \max \left[ G(x), g(x) + \alpha \sum_{y \in S} p_{xy} J^*(y) \right].$$

Furthermore, there is always an optimal policy that is stationary (i.e., of the form $\{\mu_t = \mu^*, \forall t\}$) and defined by

$$\mu^*(x) = \begin{cases} \text{stop}, & \text{if } G(x) \geq V^*(x), \\ \text{continue}, & \text{otherwise}. \end{cases}$$

Following Watkins (1989), we define the $Q$–function as the function $Q^* : S \mapsto \Re$ given by

$$Q^*(x) = g(x) + \alpha \sum_{y \in S} p_{xy} V^*(y).$$

It is easy to show that the $Q$–function uniquely satisfies

$$Q^*(x) = g(x) + \alpha \sum_{y \in S} p_{xy} \max \left[ G(y), Q^*(y) \right], \qquad \forall x \in S. \tag{1}$$

Furthermore, an optimal policy can be defined by

$$\mu^*(x) = \begin{cases} \text{stop}, & \text{if } G(x) \geq Q^*(x), \\ \text{continue}, & \text{otherwise}. \end{cases}$$

## 3   APPROXIMATING THE $Q$–FUNCTION

Classical computational approaches to solving optimal stopping problems involve computing and storing a value function in a tabular form. The most common way for doing this is through use of an iterative algorithm of the form

$$J_{k+1}(x) = \max \left[ G(x), g(x) + \alpha \sum_{y \in S} p_{xy} J_k(y) \right].$$

When the state space is extremely large, as is the typical case, two difficulties arise. The first is that computing and storing one value per state becomes intractable, and the second is that computing the summation on the right hand side becomes intractable. We will present an algorithm, motivated by Watkins' $Q$–learning, that addresses both these issues, allowing for approximate solution to optimal stopping problems with large state spaces.

## 3.1 LINEAR FUNCTION APPROXIMATORS

We consider approximations of $Q^*$ using a function of the form

$$\tilde{Q}(x,r) = \sum_{k=1}^{K} r(k)\phi_k(x).$$

Here, $r = \big(r(1),\ldots,r(K)\big)$ is a parameter vector and each $\phi_k$ is a fixed scalar function defined on the state space $S$. The functions $\phi_k$ can be viewed as basis functions (or as vectors of dimension $n$), while each $r(k)$ can be viewed as the associated weight. To approximate the $Q$–function, one usually tries to choose the parameter vector $r$ so as to minimize some error metric between the functions $\tilde{Q}(\cdot,r)$ and $Q^*(\cdot)$.

It is convenient to define a vector-valued function $\phi : S \mapsto \Re^K$, by letting $\phi(x) = \big(\phi_1(x),\ldots,\phi_K(x)\big)$. With this notation, the approximation can also be written in the form $\tilde{Q}(x,r) = (\Phi r)(x)$, where $\Phi$ is viewed as a $|S| \times K$ matrix whose $i$th row is equal to $\phi(x)$.

## 3.2 THE APPROXIMATION ALGORITHM

In the approximation scheme we propose, the Markov chain underlying the stopping problem is simulated to produce a single endless trajectory $\{x_t | t = 0, 1, 2,\ldots\}$. The algorithm is initialized with a parameter vector $r_0$, and after each time step, the parameter vector is updated according to

$$r_{t+1} = r_t + \gamma_t \phi(x_t) \Big( g(x_t) + \alpha \max \big[ \phi'(x_{t+1})r_t, G(x_{t+1}) \big] - \phi'(x_t)r_t \Big),$$

where $\gamma_t$ is a scalar stepsize.

## 3.3 CONVERGENCE THEOREM

Before stating the convergence theorem, we introduce some notation that will make the exposition more concise. Let $\pi(1),\ldots,\pi(n)$ denote the steady-state probabilities for the Markov chain. We assume that $\pi(x) > 0$ for all $x \in S$. Let $D$ be an $n \times n$ diagonal matrix with diagonal entries $\pi(1),\ldots,\pi(n)$. We define a weighted norm $\|\cdot\|_D$ by

$$\|J\|_D = \sqrt{\sum_{x \in S} \pi(x) J^2(x)}.$$

We define a "projection matrix" $\Pi$ that induces a weighted projection onto the subspace $\mathcal{X} = \{\Phi r \mid r \in \Re^K\}$ with projection weights equal to the steady-state probablilities. In particular,

$$\Pi J = \arg \min_{\bar{J} \in \mathcal{X}} \|J - \bar{J}\|_D.$$

It is easy to show that $\Pi$ is given by $\Pi = \Phi(\Phi' D \Phi)^{-1}\Phi' D$.

We define an operator $F : \Re^n \mapsto \Re^n$ by

$$FJ = g + \alpha P \max \Big[\Phi r_t, G\Big],$$

where the max denotes a componentwise maximization.

We have the following theorem that ensures soundness of the algorithm:

**Theorem 1** *Let the following conditions hold:*
*(a) The Markov chain has a unique invariant distribution $\pi$ that satisfies $\pi'P = \pi'$, with $\pi(x) > 0$ for all $x \in S$.*
*(b) The matrix $\Phi$ has full column rank; that is, the "basis functions" $\{\phi_k \mid k = 1, \ldots, K\}$ are linearly independent.*
*(c) The step sizes $\gamma_t$ are nonnegative, nonincreasing, and predetermined. Furthermore, they satisfy $\sum_{t=0}^{\infty} \gamma_t = \infty$, and $\sum_{t=0}^{\infty} \gamma_t^2 < \infty$.*
*We then have:*
*(a) The algorithm converges with probability 1.*
*(b) The limit of convergence $r^*$ is the unique solution of the equation*

$$\Pi F(\Phi r^*) = \Phi r^*.$$

*(c) Furthermore, $r^*$ satisfies*

$$\|\Phi r^* - Q^*\|_D \leq \frac{\|\Pi Q^* - Q^*\|_D}{1 - \alpha}.$$

## 3.4 OVERVIEW OF PROOF

The proof of Theorem 1 involves an analysis in a Euclidean space where the operator $F$ and projection $\Pi$ serve as tools for interpreting the algorithm's dynamics. The ideas for this type of analysis can be traced back to Van Roy and Tsitsiklis (1996) and have since been used to analyze Sutton's temporal-difference learning algorithm (Tsitsiklis and Van Roy, 1996b). Due to space limitations, we only provide an overview of the proof.

We begin by establishing that, with respect to the norm $\|\cdot\|_D$, $P$ is a nonexpansion and $F$ is a contraction. In the first case, we apply Jensen's inequality to obtain

$$
\begin{aligned}
\|PJ\|_D^2 &= \sum_{x \in S} \pi(x) \left(\sum_{y \in S} p_{xy} J(y)\right)^2 \\
&\leq \sum_{x \in S} \pi(x) \sum_{y \in S} p_{xy} J^2(y) \\
&= \sum_{y \in S} \sum_{x \in S} \pi(x) p_{xy} J^2(y) \\
&= \sum_{y \in S}^{n} \pi(y) J^2(y) \\
&= \|J\|_D^2.
\end{aligned}
$$

The fact that $F$ is a contraction now follows from the fact that

$$|(FJ)(x) - (F\bar{J})(x)| \leq \alpha|(PJ)(x) - (P\bar{J})(x)|,$$

for any $J, \bar{J} \in \Re^n$ and any state $x \in S$.

Let $s : \Re^m \mapsto \Re^m$ denote the "steady-state" expectation of the steps taken by the algorithm:

$$s(r) = E_0\Big[\phi(x_t)\left(g(x_t) + \alpha \max\left[\phi'(x_{t+1})r, G(x_{t+1})\right] - \phi'(x_t)r\right)\Big],$$

where $E_0[\cdot]$ denotes the expectation with respect to steady-state probabilities. Some simple algebra gives

$$s(r) = \Phi'D\Big(F(\Phi r) - \Phi r\Big).$$

We focus on analyzing a deterministic algorithm of the form

$$\bar{r}_{t+1} = \bar{r}_t + \gamma_t s(\bar{r}_t).$$

The convergence of the stochastic algorithm we have proposed can be deduced from that of this deterministic algorithm through use of a theorem on stochastic approximation, contained in (Benveniste, et al., 1990).

Note that the composition $\Pi F(\cdot)$ is a contraction with respect to $\| \cdot \|_D$ with contraction coefficient $\alpha$ since projection is nonexpansive and $F$ is a contraction. It follows that $\Pi F(\cdot)$ has a fixed point of the form $\Phi r^*$ for some $r^* \in \Re^m$ that uniquely satisfies

$$\Phi r^* = \Pi F(\Phi r^*).$$

To establish convergence, we consider the potential function $U(r) = \frac{1}{2}\|r - r^*\|_D^2$. We have

$$
\begin{aligned}
(\nabla U(r))'s(r) &= \left(r - r^*\right)' \Phi' D \left(F(\Phi r) - \Phi r\right) \\
&= \left(r - r^*\right)' \Phi' D \left(\Pi F(\Phi r) - (I - \Pi)F(\Phi r) - \Phi r\right) \\
&= \left(\Phi r - \Phi r^*\right)' D \left(\Pi F(\Phi r) - \Phi r\right),
\end{aligned}
$$

where the last equality follows because $\Phi' D \Pi = \Phi' D$. Using the contraction property of $F$ and the nonexpansion property of projection, we have

$$
\begin{aligned}
\|\Pi F(\Phi r) - \Phi r^*\|_D &= \|\Pi F(\Phi r) - \Pi F(\Phi r^*)\|_D \\
&\leq \alpha \|\Phi r - \Phi r^*\|_D,
\end{aligned}
$$

and it follows from the Cauchy-Schwartz inequality that

$$
\begin{aligned}
(\nabla U(r))'s(r) &= \left(\Phi r - \Phi r^*\right)' D \left(\Pi F(\Phi r) - \Phi r^* + \Phi r^* - \Phi r\right) \\
&\leq \|\Phi r - \Phi r^*\|_D \|\Pi F(\Phi r) - \Phi r^*\|_D - \|\Phi r - \Phi r^*\|_D^2 \\
&\leq (\alpha - 1)\|\Phi' r - \Phi r^*\|_D^2.
\end{aligned}
$$

Since $\Phi$ has full column rank, it follows that $(\nabla U(r))'s(r) \leq -\epsilon U(r)$, for some fixed $\epsilon > 0$, and $\bar{r}_t$ converges to $r^*$.

We can further establish the desired error bound:

$$
\begin{aligned}
\|\Phi r^* - Q^*\|_D &\leq \|\Phi r^* - \Pi Q^*\|_D + \|\Pi Q^* - Q^*\|_D \\
&= \|\Pi F(\Phi r^*) - \Pi Q^*\|_D + \|\Pi Q^* - Q^*\|_D \\
&\leq \alpha \|\Phi r^* - Q^*\|_D + \|\Pi Q^* - Q^*\|_D,
\end{aligned}
$$

and it follows that

$$\|\Phi r^* - Q^*\|_D \leq \frac{\|\Pi Q^* - Q^*\|_D}{1 - \alpha}.$$

## 4 CONCLUSION

We have proposed an algorithm for approximating $Q$-functions of optimal stopping problems using linear combinations of fixed basis functions. We have also presented a convergence theorem and overviewed the associated analysis. This paper has served a dual purpose of establishing a new methodology for solving difficult optimal stopping problems and providing a starting point for analyses of $Q$-learning-like algorithms when used in conjunction with function approximators.

The line of analysis presented in this paper easily generalizes in several directions. First, it extends to unbounded continuous state spaces. Second, it can be used to analyze certain variants of $Q$–learning that can be used for optimal stopping problems where the underlying Markov processes are not irreducible and/or aperiodic. Rigorous analyses of some extensions, as well as the case that was discussed in this paper, are presented in a forthcoming full-length paper.

## Acknowledgments

This research was supported by the NSF under grant DMI-9625489 and the ARO under grant DAAL-03-92-G-0115.

## References

Benveniste, A., Metivier, M., & Priouret, P. (1990) *Adaptive Algorithms and Stochastic Approximations*, Springer-Verlag, Berlin.

Bertsekas, D. P. (1995) *Dynamic Programming and Optimal Control*. Athena Scientific, Belmont, MA.

Gordon, G. J. (1995) Stable Function Approximation in Dynamic Programming. Technical Report: CMU-CS-95-103, Carnegie Mellon University.

Jaakkola, T., Jordan M. I., & Singh, S. P. (1994) "On the Convergence of Stochastic Iterative Dynamic Programming Algorithms," Neural Computation, Vol. 6, No. 6.

Jaakkola T., Singh, S. P., & Jordan, M. I. (1995) "Reinforcement Learning Algorithms for Partially Observable Markovian Decision Processes," in *Advances in Neural Information Processing Systems 7*, J. D. Cowan, G. Tesauro, and D. Touretzky, editors, Morgan Kaufmann.

Sutton, R. S. (1988) Learning to Predict by the Method of Temporal Differences. *Machine Learning*, 3:9-44.

Tsitsiklis, J. N. (1994) "Asynchronous Stochastic Approximation and Q-Learning," Machine Learning, vol. 16, pp. 185-202.

Tsitsiklis, J. N. & Van Roy, B. (1996a) "Feature-Based Methods for Large Scale Dynamic Programming," Machine Learning, Vol. 22, pp. 59-94.

Tsitsiklis, J. N. & Van Roy, B. (1996b) An Analysis of Temporal-Difference Learning with Function Approximation. Technical Report: LIDS-P-2322, Laboratory for Information and Decision Systems, Massachusetts Institute of Technology.

Van Roy, B. & Tsitsiklis, J. N. (1996) "Stable Linear Approximations to Dynamic Programming for Stochastic Control Problems with Local Transitions," in *Advances in Neural Information Processing Systems 8*, D. S. Touretzky, M. C. Mozer, and M. E. Hasselmo, editors, MIT Press.

Watkins, C. J. C. H. (1989) Learning from Delayed Rewards. Doctoral dissertation, University of Cambridge, Cambridge, United Kingdom.

Watkins, C. J. C. H. & Dayan, P. (1992) "Q–learning," Machine Learning, vol. 8, pp. 279-292.